# Regularized Boost for Semi-Supervised Learning

**Ke Chen and Shihai Wang**
School of Computer Science
The University of Manchester
Manchester M13 9PL, United Kingdom
{chen,swang}@cs.manchester.ac.uk

## Abstract

Semi-supervised inductive learning concerns how to learn a decision rule from a data set containing both labeled and unlabeled data. Several boosting algorithms have been extended to semi-supervised learning with various strategies. To our knowledge, however, none of them takes local smoothness constraints among data into account during ensemble learning. In this paper, we introduce a local smoothness regularizer to semi-supervised boosting algorithms based on the universal optimization framework of margin cost functionals. Our regularizer is applicable to existing semi-supervised boosting algorithms to improve their generalization and speed up their training. Comparative results on synthetic, benchmark and real world tasks demonstrate the effectiveness of our local smoothness regularizer. We discuss relevant issues and relate our regularizer to previous work.

## 1 Introduction

Semi-supervised inductive learning concerns the problem of automatically learning a decision rule from a set of both labeled and unlabeled data, which has received a great deal of attention due to enormous demands of real world learning tasks ranging from data mining to medical diagnosis [1]. From different perspectives, a number of semi-supervised learning algorithms have been proposed [1],[2], e.g., self-training, co-training, generative models along with the EM algorithm, transductive learning models and graph-based methods.

In semi-supervised learning, the ultimate goal is to find out a classification function which not only minimizes classification errors on the labeled training data but also must be compatible with the input distribution by inspecting their values on unlabeled data. To work towards the goal, unlabeled data can be exploited to discover how data is distributed in the input space and then the information acquired from the unlabeled data is used to find a good classifier. As a generic framework, regularization has been used in semi-supervised learning to exploit unlabeled data by working on well known semi-supervised learning assumptions, i.e., the smoothness, the cluster, and the manifold assumptions [1], which leads to a number of regularizers applicable to various semi-supervised learning paradigms, e.g., the measure-based [3], the manifold-based [4], the information-based [5], the entropy-based [6], harmonic mixtures [7] and graph-based regularization [8].

As a generic ensemble learning framework [9] , boosting works by sequentially constructing a linear combination of base learners that concentrate on difficult examples, which results in a great success in supervised learning. Recently boosting has been extended to semi-supervised learning with different strategies. Within the universal optimization framework of margin cost functional [9], semi-supervised MarginBoost [10] and ASSEMBLE [11] were proposed by introducing the "pseudo-classes" to unlabeled data for characterizing difficult unlabeled examples. In essence, such extensions work in a self-training way; the unlabeled data are assigned pseudo-class labels based on the constructed ensemble learner so far, and in turn the pseudo-class labels achieved will be used to find out a new proper learner to be added to the ensemble. The co-training idea was extended to

boosting, e.g. CoBoost [12]. More recently, the Agreement Boost algorithm [13] has been developed with a theoretic justification of benefits from the use of multiple boosting learners within the co-training framework. To our knowledge, however, none of the aforementioned semi-supervised boosting algorithms has taken the local smoothness constraints into account.

In this paper, we exploit the local smoothness constraints among data by introducing a regularizer to semi-supervised boosting. Based on the universal optimization framework of margin cost functional for boosting [9], our regularizer is applicable to existing semi-supervised boosting algorithms [10]-[13]. Experimental results on the synthetic, benchmark and real world classification tasks demonstrate its effectiveness of our regularizer in semi-supervised boosting learning.

In the reminder of this paper, Sect. 2 briefly reviews semi-supervised boosting learning and presents our regularizer. Sect. 3 reports experimental results and the behaviors of regularized semi-supervised boosting algorithms. Sect. 4 discusses relevant issues and the last section draws conclusions.

## 2 Semi-supervised boosting learning and regularization

In the section, we first briefly review the basic idea behind existing semi-supervised boosting algorithms within the universal optimization framework of margin cost functional [9] for making it self-contained. Then we present our Regularized Boost based on the previous work.

### 2.1 Semi-supervised boosting learning

Given a training set, $S = L \cup U$, of $|L|$ labeled examples, $L = \{(\mathbf{x}_1, y_1), \cdots, (\mathbf{x}_{|L|}, y_{|L|})\}$, and $|U|$ unlabeled examples, $U = \{\mathbf{x}_{|L|+1}, \cdots, \mathbf{x}_{|L|+|U|}\}$, we wish to construct an ensemble learner $F(\mathbf{x}) = \sum_t w_t f_t(\mathbf{x})$, where $w_t$ is coefficients for linear combination and $f_t(\mathbf{x})$ is a base learner, so that $P(F(\mathbf{x}) \neq y)$ is small. Since there exists no label information available for unlabeled data, the critical idea underlying semi-supervised boosting is introducing a pseudo-class [11] or a pseudo margin [10] concept within the universal optimization framework [9] to unlabeled data. Similar to an approach in supervised learning, e.g., [14], a multi-class problem can be converted into binary classification forms. Therefore, our presentation below focuses on the binary classification problem only; i.e. $y \in \{-1, 1\}$. The pseudo-class of an unlabeled example, $\mathbf{x}$, is typically defined as $y = \text{sign}[F(\mathbf{x})]$ [11] and its corresponding pseudo margin is $yF(\mathbf{x}) = |F(\mathbf{x})|$ [10],[11].

Within the universal optimization framework of margin cost functional [9], the semi-supervised boosting learning is to find $F$ such that the cost of functional

$$\mathcal{C}(F) = \sum_{\mathbf{x}_i \in L} \alpha_i C[y_i F(\mathbf{x}_i)] + \sum_{\mathbf{x}_i \in U} \alpha_i C[|F(\mathbf{x}_i)|] \tag{1}$$

is minimized for some non-negative and monotonically decreasing cost function $C : \mathbb{R} \to \mathbb{R}$ and the weight $\alpha_i \in \mathbb{R}^+$. In the universal optimization framework [9], constructing an ensemble learner needs to choose a base learner, $f(\mathbf{x})$, to maximize the inner product $-\langle \nabla \mathcal{C}(F), f \rangle$. For unlabeled data, a subgradient of $\mathcal{C}(F)$ in (1) has been introduced to tackle its non-differentiable problem [11] and then unlabeled data of pseudo-class labels can be treated in the same way as labeled data in the optimization problem. As a result, finding a proper $f(\mathbf{x})$ amounts to maximizing

$$-\langle \nabla \mathcal{C}(F), f \rangle = \sum_{i: f(\mathbf{x}_i) \neq y_i} \alpha_i C'[y_i F(\mathbf{x}_i)] \quad - \sum_{i: f(\mathbf{x}_i) = y_i} \alpha_i C'[y_i F(\mathbf{x}_i)], \tag{2}$$

where $y_i$ is the true class label if $\mathbf{x}_i$ is a labeled example or a pseudo-class label otherwise. After dividing through by $-\sum_{i \in S} \alpha_i C'[y_i F(\mathbf{x}_i)]$ on both sides of (2), finding $f(\mathbf{x})$ to maximize $-\langle \nabla \mathcal{C}(F), f \rangle$ is equivalent to searching for $f(\mathbf{x})$ to minimize

$$\sum_{i: f(\mathbf{x}_i) \neq y_i} D(i) \quad - \sum_{i: f(\mathbf{x}_i) = y_i} D(i) = 2 \sum_{i: f(\mathbf{x}_i) \neq y_i} D(i) \; - \; 1, \tag{3}$$

where $D(i)$, for $1 \leq i \leq |L| + |U|$, is the empirical data distribution defined as $D(i) = \frac{\alpha_i C'[y_i F(\mathbf{x}_i)]}{\sum_{i \in S} \alpha_i C'[y_i F(\mathbf{x}_i)]}$. From (3), a proper base learner, $f(\mathbf{x})$, can be found by minimizing weighted errors $\sum_{i: f(\mathbf{x}_i) \neq y_i} D(i)$. Thus, any boosting algorithms specified for supervised learning [9] are now applicable to semi-supervised learning with the aforementioned treatment.

For co-training based semi-supervised boosting algorithms [12],[13], the above semi-supervised boosting procedure is applied to each view of data to build up a component ensemble learner. Instead of self-training, the pseudo-class label of an unlabeled example for a specific view is determined by ensemble learners trained on other views of this example. For example, the Agreement Boost [13] defines the co-training cost functional as

$$\mathcal{C}(F^1, \cdots, F^J) = \sum_{j=1}^{J} \sum_{\mathbf{x}_i \in L} C[y_i F^j(\mathbf{x}_i)] + \eta \sum_{\mathbf{x}_i \in U} C[-V(\mathbf{x}_i)]. \qquad (4)$$

Here $J$ views of data are used to train $J$ ensemble learners, $F^1, \cdots, F^J$, respectively. The disagreement of $J$ ensemble learners for an unlabeled example, $\mathbf{x}_i \in U$, is $V(\mathbf{x}_i) = \frac{1}{J} \sum_{j=1}^{J} [F^j(\mathbf{x}_i)]^2 - \left[\frac{1}{J} \sum_{j=1}^{J} F^j(\mathbf{x}_i)\right]^2$ and the weight $\eta \in \mathbb{R}^+$. In light of view $j$, the pseudo-class label of an unlabeled example, $\mathbf{x}_i$, is determined by $y_i = \text{sign}\left[\frac{1}{J} \sum_{j=1}^{J} F^j(\mathbf{x}_i) - F^j(\mathbf{x}_i)\right]$. Thus, the minimization of (3) with such pseudo-class labels leads to a proper base learner $f^j(\mathbf{x})$ to be added to $F^j(\mathbf{x})$.

## 2.2  Boosting with regularization

Motivated by the work on the use of regularization in semi-supervised learning [3]-[8], we introduce a local smoothness regularizer to semi-supervised boosting based on the universal optimization framework of margin cost functional [9], which results in a novel objective function:

$$\mathcal{T}(F, f) = -\langle \nabla \mathcal{C}(F), f \rangle - \sum_{i:\mathbf{x}_i \in S} \beta_i R(i), \qquad (5)$$

where $\beta_i \in \mathbb{R}^+$ is a weight, determined by the input distribution to be discussed in Sect. 4, associated with each training example and the local smoothness around an example, $\mathbf{x}_i$, is measured by

$$R(i) = \sum_{j:\mathbf{x}_j \in S, j \neq i} W_{ij} \tilde{C}(-I_{ij}). \qquad (6)$$

Here, $I_{ij}$ is a class label compatibility function for two different examples $\mathbf{x}_i, \mathbf{x}_j \in S$ and defined as $I_{ij} = |y_i - y_j|$ where $y_i$ and $y_j$ are the true labels of $\mathbf{x}_i$ and $\mathbf{x}_j$ for labeled data or their pseudo-class labels otherwise. $\tilde{C} : \mathbb{R} \to \mathbb{R}$ is a monotonically decreasing function derived from the cost function adopted in (1) so that $\tilde{C}(0) = 0$. $W_{ij}$ is an affinity measure defined by $W_{ij} = \exp(-||\mathbf{x}_i - \mathbf{x}_j||^2/2\sigma^2)$ where $\sigma$ is a bandwidth parameter. To find a proper base learner, $f(\mathbf{x})$, we now need to maximize $\mathcal{T}(F, f)$ in (5) so as to minimize not only misclassification errors as before (see Sect. 2.1) but also the local class label incompatibility cost for smoothness.

In order to use the objective function in (5) for boosting learning, we need to have the new empirical data distribution and the termination condition. Inserting (2) into (5) results in

$$\mathcal{T}(F, f) = \sum_{i:f(\mathbf{x}_i) \neq y_i} \alpha_i C'[y_i F(\mathbf{x}_i)] - \sum_{i:f(\mathbf{x}_i) = y_i} \alpha_i C'[y_i F(\mathbf{x}_i)] - \sum_{i:\mathbf{x}_i \in S} \beta_i R(i). \qquad (7)$$

Since an appropriate cost function used in (1) is non-negative and monotonically decreasing, $C'[y_i F(\mathbf{x}_i)]$ is always negative and $R(i)$ is non-negative according to its definition in (6). Therefore, we can define our empirical data distribution as

$$\tilde{D}(i) = \frac{\alpha_i C'[y_i F(\mathbf{x}_i)] - \beta_i R(i)}{\sum_{k:\mathbf{x}_k \in S} \left\{ \alpha_k C'[y_k F(\mathbf{x}_k)] - \beta_k R(k) \right\}}, \quad 1 \leq i \leq |L| + |U|. \qquad (8)$$

$\tilde{D}(i)$ is always non-negative based on definitions of cost function in (1) and $R(i)$ in (6). Applying (8) to (7) with some mathematical development similar to that described in Sect. 2.1, we can show that finding a proper base learner $f(\mathbf{x})$ to maximize $\mathcal{T}(F, f)$ is equivalent to finding $f(\mathbf{x})$ to minimize

$$\sum_{i:f(\mathbf{x}_i) \neq y_i} \tilde{D}(i) - \sum_{i:f(\mathbf{x}_i) = y_i} \tilde{D}(i) - 2 \sum_{i:f(\mathbf{x}_i) = y_i} \frac{\beta_i R(i)}{\sum_{k:\mathbf{x}_k \in S} \left\{ \alpha_k C'[y_k F(\mathbf{x}_k)] - \beta_k R(k) \right\}},$$

which is equal to

$$2 \underbrace{\sum_{i:f(\mathbf{x}_i) \neq y_i} \tilde{D}(i)}_{\text{misclassification errors}} + 2 \underbrace{\sum_{i:f(\mathbf{x}_i) = y_i} \frac{-\beta_i R(i)}{\sum_{k:\mathbf{x}_k \in S} \left\{ \alpha_k C'[y_k F(\mathbf{x}_k)] - \beta_k R(k) \right\}}}_{\text{local class label incompatibility}} - 1. \qquad (9)$$

In (9), the first term refers to misclassification errors while the second term corresponds to the class label incompatibility of a data point with its nearby data points even though this data point itself fits well. In contrast to (3), finding a proper base learner, $f(\mathbf{x})$, now needs to minimize not only the misclassification errors but also the local class label incompatibility in our Regularized Boost. Accordingly, a new termination condition of our Regularized Boost is derived from (9) as $\epsilon \geq \frac{1}{2}$ where $\epsilon = \sum_{i:f(\mathbf{x}_i) \neq y_i} \tilde{D}(i) + \sum_{i:f(\mathbf{x}_i)=y_i} \frac{-\beta_i R(i)}{\sum_{k:\mathbf{x}_k \in S} \left\{ \alpha_k C'[y_k F(\mathbf{x}_k)] - \beta_k R(k) \right\}}$.

Once finding an optimal base learner, $f_{t+1}(\mathbf{x})$, at step $t+1$, we need to choose a proper weight, $w_{t+1}$, to form a new ensemble, $F_{t+1}(\mathbf{x}) = F_t(\mathbf{x}) + w_{t+1} f_{t+1}(\mathbf{x})$. In our Regularized Boost, we choose $w_{t+1} = \frac{1}{2} \log \left( \frac{1-\epsilon}{\epsilon} \right)$ by simply treating pseudo-class labels for unlabeled data as same as true labels of labeled data, as suggested in [11].

## 3 Experiments

In this section, we report experimental results on synthetic, benchmark and real data sets. Although our regularizer is applicable to existing semi-supervised boosting [10]-[13], we mainly apply it to the ASSEMBLE [11], a winning algorithm from the NIPS 2001 Unlabeled Data Competition, on a variety of classification tasks. In addition, our regularizer is also used to train component ensemble learners of the Agreement Boost [13] for binary classification benchmark tasks since the algorithm [13] in its original form can cope with binary classification only. In our experiments, we use $C(\gamma) = e^{-\gamma}$ in (1) and $\tilde{C}(\gamma) = C(\gamma) - 1$ in (6) and set $\alpha_i = 1$ in (1) and $\beta_i = \frac{1}{2}$ in (5).

For synthetic and benchmark data sets, we always randomly select 20% of examples as testing data except that a benchmark data set has pre-defined a training/test split. Accordingly, the remaining examples used as a training set or those in a pre-defined training set, $S$, are randomly divided into two subsets, i.e., labeled data ($L$) and unlabeled data ($U$), and the ratio between labeled and unlabeled data is 1:4 in our experiments. For reliability, each experiment is repeated for ten times. To test the effectiveness of our Regularized Boost across different base learners, we perform all experiments with $K$ nearest-neighbors (KNN) classifier, a local classifier, and multi-layer perceptron (MLP), a global classifier, where 3NN and a single hidden layered MLP are used in our experiments. For comparison, we report results of a semi-supervised boosting algorithm (i.e., ASSEMBLE [11] or Agreement Boost [13]) and its regularized version (i.e., Regularized Boost). In addition, we also provide results of a variant of Adaboost [14] trained on the labeled data only for reference. The above experimental method conforms to those used in semi-supervised boosting methods [10]-[13] as well as other empirical studies of semi-supervised learning methods, e.g., [15].

### 3.1 Synthetic data set

We use a Gaussian mixture model of four components to generate a data set of four categories in the 2-D space; 200 examples are in each category, as illustrated in Figure 1(a). We wish to test our regularizer on this intuitive multi-class classification task of a high optimal Bayes error.

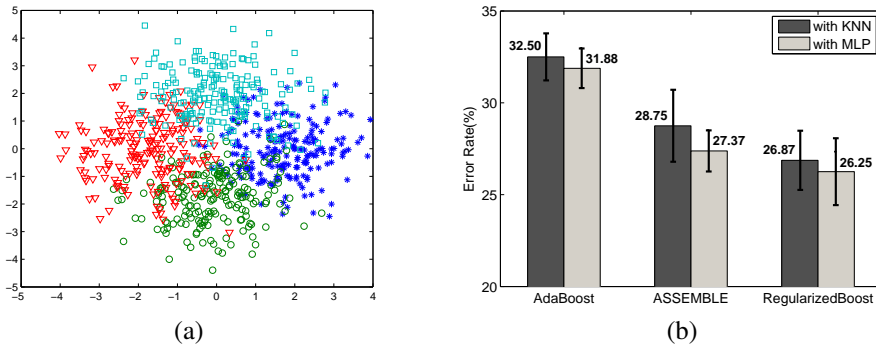

(a)      (b)

Figure 1: Synthetic data classification task. (a) The data set. (b) Classification results

From Figure 1(b), it is observed that the use of unlabeled data improves the performance of Adaboost and the use of our regularizer further improves the generalization performance of the ASSEMBLE

by achieving an averaging error rate closer to the optimal Bayes error no matter what kind of a base learner is used. Our further observation via visualization with the ground truth indicates that the use of our regularizer leads to smoother decision boundaries than the original ASSEMBLE, which yields the better generalization performance.

## 3.2   Benchmark data sets

To assess the performance of our regularizer for semi-supervised boosting algorithms, we perform a series of experiments on benchmark data sets from the UCI machine learning repository [16] without any data transformation. In our experiments, we use the same initialization conditions for all boosting algorithms. Our empirical work suggests that a maximum number of 100 boosting steps is sufficient to achieve the reasonable performance for those benchmark tasks. Hence, we set such a maximum number of boosting steps to stop all boosting algorithms for a sensible comparison.

We first apply our regularizer to the ASSEMBLE [11] on five UCI benchmark classification tasks of different categories[16]: BUPA liver disorders (BUPA), Wisconsin Diagnostic Breast Cancer (WDBC), Balance Scale Weight & Distance (BSWD), Car Evaluation Database (CAR), and Optical Recognition of Handwritten Digits (OPTDIGITS) where its data set has been split into the fixed training and testing subsets in advance by the data collector.

Table 1: Error rates (mean±dev.)% of AdaBoost, ASSEMBLE and Regularized Boost (RegBoost) with different base learners on five UCI classification data sets.

| Data Set | KNN | | | MLP | | |
|---|---|---|---|---|---|---|
| | AdaBoost | ASSEMBLE | RegBoost | AdaBoost | ASSEMBLE | RegBoost |
| BUPA | 37.7±3.4 | 36.1±3.0 | 34.9±3.1 | 35.1±1.1 | 31.2±6.7 | 28.8±5.6 |
| WDBC | 8.3±1.9 | 4.1±1.0 | 3.7±2.0 | 9.7±2.0 | 3.5±0.9 | 3.2±0.8 |
| BSWD | 22.2±0.9 | 18.7±0.4 | 17.4±0.9 | 16.8±2.8 | 14.4±2.4 | 13.6±2.6 |
| CAR | 31.3±1.2 | 24.4±0.7 | 23.2 ±1.1 | 30.6±3.0 | 20.5±0.9 | 17.7±1.1 |
| OTIDIGITS | 4.9±0.1 | 3.1±0.5 | 2.7±0.7 | 6.3±0.2 | 5.2±0.2 | 5.0±0.2 |

Table 1 tabulates the results of different boosting learning algorithms. It is evident from Table 1 that in general the use of unlabeled data constantly improves the generalization performance in contrast to the performance of AdaBoost and the use of our regularizer in the ASSEMBLE always further reduces its error rates on all five data sets no matter what kind of a base learner is used. It is also observed that the use of different base learners results in various performance on five data sets; the use of KNN as a base learner yields better performance on the WDBC and OPTDIGITS data set whereas the use of MLP as a base learner outperforms its KNN counterpart on other three data sets. Apparently the nature of a base learner, e.g., global vs. local classifiers, may determine if it is suitable for a classification task. It is worth mentioning that for the OPTDIGITS data set the lowest error rate achieved by 3NN with the entire training set, i.e., using all 3823 examples as training prototypes, is around 2.2% on the testing set, as reported in the literature [16]. In contrast, the ASSEMBLE [11] on 3NN equipped with our regularizer yields an error rate of 2.7% on average despite the fact that our Regularized Boost algorithm simply uses 765 labeled examples.

Table 2: Error rates (mean±dev.)% of AdaBoost, Agreement Boost and Regularized Boost (RegBoost) on five UCI binary classification data sets.

| Data Set | AdaBoost-KNN | AdaBoost-MLP | AgreementBoost | RegBoost |
|---|---|---|---|---|
| BUPA | 37.7±3.4 | 35.1±1.1 | 30.4±7.5 | 28.9±5.8 |
| WDBC | 8.3±1.9 | 9.7±2.0 | 3.3±0.7 | 3.0±0.8 |
| VOTE | 9.0±1.5 | 10.6±0.5 | 4.4±0.8 | 2.8±0.6 |
| AUSTRALIAN | 37.7±1.2 | 21.0±3.4 | 16.7±2.1 | 15.2±2.8 |
| KR-vs-KP | 15.6±0.7 | 7.1±0.2 | 6.3±1.3 | 5.2±1.6 |

We further apply our regularizer to the Agreement Boost [13]. Due to the limitation of this algorithm [13], we can use only the binary classification data sets to test the effectiveness. As a result, we use BUPA and WDBC mentioned above and three additional UCI binary classification data sets [16]: 1984 U.S. Congressional Voting Records (VOTE), Australian Credit Approval (AUSTRALIAN)

and Chess End-Game King Rook versus King Pawn (KR-vs-KP). As required by the Agreement Boost [13], the KNN and the MLP classifiers as base learners are used to construct two component ensemble learners without and with the use of our regularizer in experiments, which corresponds to its original and regularized version of the Agreement Boost.

Table 2 tabulates results produced by different boosting algorithms. It is evident from Table 2 that the use of our regularizer in its component ensemble learners always leads the Agreement Boost to improve its generalization on five benchmark tasks while its original version trained with labeled and unlabeled data considerably outperforms the Adaboost trained with labeled data only.

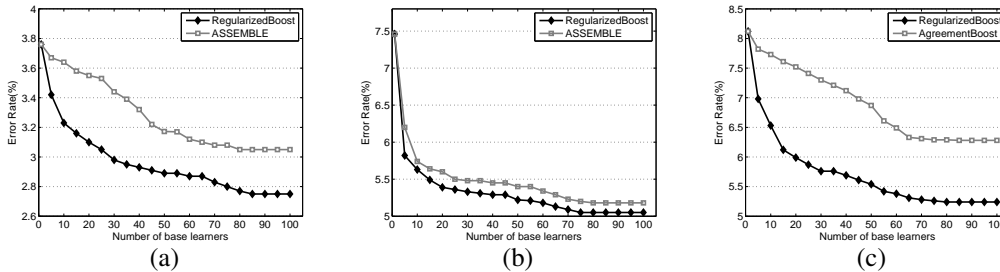

(a)  (b)  (c)

Figure 2: Behaviors of semi-supervised boosting algorithms: the original version vs. the regularized version. (a) The ASSEMBLE with KNN on the OPTDIGITS. (b) The ASSEMBLE with MLP on the OPTDIGITS. (c) The Agreement Boost on the KR-vs-KP.

We investigate behaviors of regularized semi-supervised boosting algorithms on two largest data sets, OPTDIGITS and VR-vs-VP. Figure 2 shows the averaging generalization performance achieved by stopping a boosting algorithm at different boosting steps. From Figure 2, the use of our regularizer in the ASSEMBLE regardless of base learners adopted and the Agreement Boost always yields fast training. As illustrated in Figures 2(a) and 2(b), the regularized version of the ASSEMBLE with KNN and MLP takes only 22 and 46 boosting steps on average to reach the performance of the original ASSEMBLE after 100 boosting steps, respectively. Similarly, Figure 2(c) shows that the regularized Agreement Boost takes only 12 steps on average to achieve the performance of its original version after 100 boosting steps.

## 3.3 Facial expression recognition

Facial expression recognition is a typical semi-supervised learning task since labeling facial expressions is an extremely expensive process and very prone to errors due to ambiguities. We test the effectiveness of our regularizer by using a facial expression benchmark database, JApanese Female Facial Expression (JAFFE) [17] where there are 10 female expressers who posed 3 or 4 examples for each of seven universal facial expressions (anger, disgust, fear, joy, neutral, sadness and surprise), as exemplified in Figure 3(a), and 213 pictures of $256 \times 256$ pixels were collected totally.

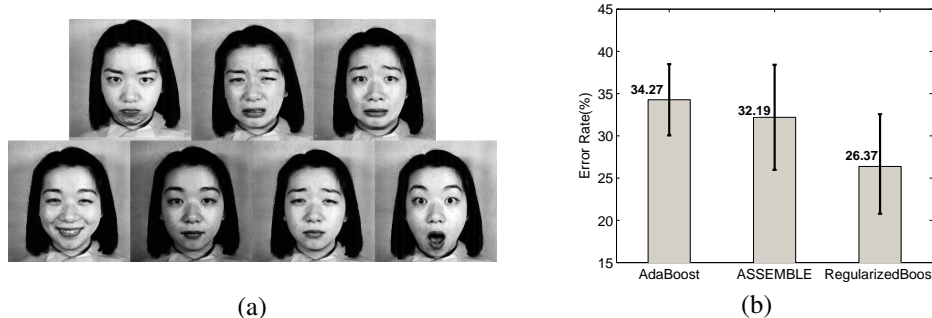

(a)  (b)

Figure 3: Facial expression recognition on the JAFFE. (a) Exemplar pictures corresponding to seven universal facial expressions. (b) Classification results of different boosting algorithms.

In our experiments, we first randomly choose $20\%$ images (balanced to seven classes) as testing data and the rest of images constitute a training set ($S$) randomly split into labeled ($L$) and unlabeled ($U$) data of equal size in each trial. We apply the independent component analysis and then the

principal component analysis (PCA) to each image for feature extraction and use only first 40 PCA coefficients to form a feature vector. A single hidden layered MLP of 30 hidden neurons is used as the based learner. We set a maximum number of 1000 boosting rounds to stop the algorithms if their termination conditions are not met while the same initialization is used for all boosting algorithms. For reliability, the experiment is repeated 10 times. From Figure 3(b), it is evident that the ASSEMBLE with our regularizer yields $5.82\%$ error reduction on average; an averaging error rate of $26.37\%$ achieved is even better than that of some supervised learning methods on the same database, e.g., [18] where around $70\%$ images were used to train a convolutional neural network and an averaging error rate of $31.5\%$ was achieved on the remaining images.

## 4 Discussions

In this section, we discuss issues concerning our regularizer and relate it to previous work in the context of regularization in semi-supervised learning.

As defined in (5), our regularizer has a parameter, $\beta_i$, associated with each training point, which can be used to encode the information of the marginal or input distribution, $P(\mathbf{x})$, by setting $\beta_i = \lambda P(\mathbf{x})$ where $\lambda$ is a tradeoff or regularization parameter. Thus, the use of $\beta_i$ would make the regularization take effect only in dense regions although our experiments reported were carried out by setting $\beta_i = \frac{1}{2}$; i.e., we were using a weak assumption that data are scattered uniformly throughout the whole space. In addition, (6) uses an affinity metric system to measure the proximity of data points and can be extended by incorporating the manifold information, if available, into our regularizer.

Our local smoothness regularizer plays an important role in re-sampling all training data including labeled and unlabeled data for boosting learning. As uncovered in (9), the new empirical distribution based on our regularizer not only assigns a large probability to a data point misclassified but also may cause a data point even classified correctly in the last round of boosting learning but located in a "non-smoothing" region to be assigned a relatively large probability, which distinguishes our approach from existing boosting algorithms where the distribution for re-sampling training data is determined solely by misclassification errors. For unlabeled data, such an effect always makes sense to work on the smoothness and the cluster assumptions [1] as performed by existing regularization techniques [3]-[8]. For labeled data, it actually has an effect that the labeled data points located in a "non-smoothing" region is more likely to be retained in the next round of boosting learning. As exemplified in Figure 1, such points are often located around boundaries between different classes and therefore more informative in determining a decision boundary, which would be another reason why our regularizer improves the generalization of semi-supervised boosting algorithms.

The use of manifold smoothness in a special form of Adaboost, *marginal* Adaboost, has been attempted in [19] where the graph Laplacian regularizer was applied to select base learners by the adaptive penalization of base learners according to their decision boundaries and the actual manifold structural information. In essence, the objective of using manifold smoothness in our Regularized Boost is identical to theirs in [19] but we accomplish it in a different way. We encode the manifold smoothness into the empirical data distribution used in boosting algorithms for semi-supervised learning, while their implementation adaptively adjusts the edge offset in the marginal Adaboost algorithm for a weight decay used in the linear combination of based learners [19]. In contrast, our implementation is simpler yet applicable to any boosting algorithms for semi-supervised learning, while theirs needs to be fulfilled via the marginal Adaboost algorithm even though their regularized marginal Adaboost is applicable to both supervised and semi-supervised learning indeed.

By comparison with existing regularization techniques used in semi-supervised learning, our Regularized Boost is closely related to graph-based semi-supervised learning methods, e.g., [8]. In general, a graph-based method wants to find a function to simultaneously satisfy two conditions [2]: *a*) it should be close to given labels on the labeled nodes, and *b*) it should be smooth on the whole graph. In particular, the work in [8] develops a regularization framework to carry out the above idea by defining the global and local consistency terms in their cost function. Similarly, our cost function in (9) has two terms explicitly corresponding to global and local consistency though true labels of labeled data never change during our boosting learning, which resembles theirs [8]. Nevertheless, a graph-based algorithm is an iterative label propagation process on a graph where a regularizer directly gets involved in label modification over the graph, whereas our Regularized Boost is an iterative process that runs a base learner on various distributions over training data where

our regularizer simply plays a role in determining distributions. In general, a graph-based algorithm is applicable to transductive learning only although it can be combined with other methods, e.g. a mixture model [7], for inductive learning. In contrast, our Regularized Boost is developed for inductive learning. Finally it is worth stating that unlike most of existing regularization techniques used in semi-supervised learning, e.g., [5],[6], our regularization takes effect on both labeled and unlabeled data while theirs are based on unlabeled data only.

## 5 Conclusions

We have proposed a local smoothness regularizer for semi-supervising boosting learning and demonstrated its effectiveness on different types of data sets. In our ongoing work, we are working for a formal analysis to justify the advantage of our regularizer and explain the behaviors of Regularized Boost, e.g. fast training, theoretically.

**References**

[1] Chapelle, O., Schölkopf, B., & Zien, A. (2006) *Semi-Supervised Learning*. Cambridge, MA: MIT Press.

[2] Zhu, X. (2006) Semi-supervised learning literature survey. Computer Science TR-1530, University of Wisconsin - Madison, U.S.A.

[3] Bousquet, O., Chapelle, O., & Hein, M. (2004) Measure based regularization. In *Advances in Neural Information Processing Systems 16*. Cambridge, MA: MIT Press.

[4] Belkin, M., Niyogi, P., & Sindhwani, V. (2004) Manifold regularization: a geometric framework for learning from examples. Technical Report, University of Michigan, U.S.A.

[5] Szummer, M., & Jaakkola, T. (2003) Information regularization with partially labeled data. In *Advances in Neural Information Processing Systems 15*. Cambridge, MA: MIT Press.

[6] Grandvalet, Y., & Begio, Y. (2005) Semi-supervised learning by entropy minimization. In *Advances in Neural Information Processing Systems 17*. Cambridge, MA: MIT Press.

[7] Zhu, X., & Lafferty, J. (2005) Harmonic mixtures: combining mixture models and graph-based methods for inductive and scalable semi-supervised learning. In *Proc. Int. Conf. Machine Learning*, pp. 1052-1059.

[8] Zhou, D., Bousquet, O., Lal, T., Weston, J., & Scḧlkopf, B. (2004) Learning with local and global consistency. In *Advances in Neural Information Processing Systems 16*. Cambridge, MA: MIT Press.

[9] Mason, L., Bartlett, P., Baxter, J., & Frean, M. (2000) Functional gradient techniques for combining hypotheses. In *Advances in Large Margin Classifiers*. Cambridge, MA: MIT Press.

[10] d'Alché-Buc, F., Grandvalet, Y., & Ambroise, C. (2002) Semi-supervised MarginBoost. In *Advances in Neural Information Processing Systems 14*. Cambridge, MA: MIT Press.

[11] Bennett, K., Demiriz, A., & Maclin, R. (2002) Expoliting unlabeled data in ensemble methods. In *Proc. ACM Int. Conf. Knowledge Discovery and Data Mining*, pp. 289-296.

[12] Collins, M., & Singer, Y. (1999) Unsupervised models for the named entity classification. In *Proc. SIGDAT Conf. Empirical Methods in Natural Language Processing and Very Large Corpora*.

[13] Leskes, B. (2005) The value of agreement, a new boosting algorithm. In *Proc. Int. Conf. Algorithmic Learning Theory (LNAI 3559)*, pp. 95-110, Berlin: Springer-Verlag.

[14] Günther, E., & Pfeiffer, K.P. (2005) Multiclass boosting for weak classifiers. *Journal of Machine Learning Research* **6**:189-210.

[15] Nigam, K., McCallum, A., Thrum, S., & Mitchell, T. (2000) Using EM to classify text from labeled and unlabeled documents. *Machine Learning* **39**:103-134.

[16] Blake, C., Keogh, E., & Merz, C.J. (1998) UCI repository of machine learning databases. University of California, Irvine. [on-line] http://www.ics.uci.edu/ mlearn/MLRepository.html

[17] The JAFFE Database. [Online] http://www.kasrl.org/jaffe.html

[18] Fasel, B. (2002) Robust face analysis using convolutional neural networks. In *Proc. Int. Conf. Pattern Recognition*, vol. 2, pp. 40-43.

[19] Kégl, B., & Wang, L. (2004) Boosting on manifolds: adaptive regularization of base classifier. In *Advances in Neural Information Processing Systems 16*. Cambridge, MA: MIT Press.
